# Robust Neural Network Regression for Offline and Online Learning

**Thomas Briegel***
Siemens AG, Corporate Technology
D-81730 Munich, Germany
*thomas.briegel@mchp.siemens.de*

**Volker Tresp**
Siemens AG, Corporate Technology
D-81730 Munich, Germany
*volker.tresp@mchp.siemens.de*

## Abstract

We replace the commonly used Gaussian noise model in nonlinear regression by a more flexible noise model based on the Student-$t$-distribution. The degrees of freedom of the $t$-distribution can be chosen such that as special cases either the Gaussian distribution or the Cauchy distribution are realized. The latter is commonly used in robust regression. Since the $t$-distribution can be interpreted as being an infinite mixture of Gaussians, parameters and hyperparameters such as the degrees of freedom of the $t$-distribution can be learned from the data based on an EM-learning algorithm. We show that modeling using the $t$-distribution leads to improved predictors on real world data sets. In particular, if outliers are present, the $t$-distribution is superior to the Gaussian noise model. In effect, by adapting the degrees of freedom, the system can "learn" to distinguish between outliers and non-outliers. Especially for online learning tasks, one is interested in avoiding inappropriate weight changes due to measurement outliers to maintain stable online learning capability. We show experimentally that using the $t$-distribution as a noise model leads to stable online learning algorithms and outperforms state-of-the art online learning methods like the extended Kalman filter algorithm.

## 1 INTRODUCTION

A commonly used assumption in nonlinear regression is that targets are disturbed by independent additive Gaussian noise. Although one can derive the Gaussian noise assumption based on a maximum entropy approach, the main reason for this assumption is practicability: under the Gaussian noise assumption the maximum likelihood parameter estimate can simply be found by minimization of the squared error. Despite its common use it is far from clear that the Gaussian noise assumption is a good choice for many practical problems. A reasonable approach therefore would be a noise distribution which contains the Gaussian as a special case but which has a tunable parameter that allows for more flexible distributions. In this paper we use the Student-$t$-distribution as a noise model which contains two free parameters – the degrees of freedom $\nu$ and a width parameter $\sigma^2$. A nice feature of the $t$-distribution is that if the degrees of freedom $\nu$ approach infinity, we recover the Gaussian noise model. If $\nu < \infty$ we obtain distributions which are more heavy-tailed than the Gaussian distribution including the Cauchy noise model with $\nu = 1$. The latter

is commonly used for robust regression. The first goal of this paper is to investigate if the additional free parameters, e.g. $\nu$, lead to better generalization performance for real world data sets if compared to the Gaussian noise assumption with $\nu = \infty$. The most common reason why researchers depart from the Gaussian noise assumption is the presence of outliers. Outliers are errors which occur with low probability and which are not generated by the data-generation process that is subject to identification. The general problem is that a few (maybe even one) outliers of high leverage are sufficient to throw the standard Gaussian error estimators completely off-track (Rousseeuw & Leroy, 1987). In the second set of experiments we therefore compare how the generalization performance is affected by outliers, both for the Gaussian noise assumption and for the $t$-distribution assumption. Dealing with outliers is often of critical importance for online learning tasks. Online learning is of great interest in many applications exhibiting non-stationary behavior like tracking, signal and image processing, or navigation and fault detection (see, for instance the NIPS*98 Sequential Learning Workshop). Here one is interested in avoiding inappropriate weight chances due to measurement outliers to maintain stable online learning capability. Outliers might result in highly fluctuating weights and possible even instability when estimating the neural network weight vector online using a Gaussian error assumption. State-of-the art online algorithms like the extended Kalman filter, for instance, are known to be nonrobust against such outliers (Meinhold & Singpurwalla, 1989) since they are based on a Gaussian output error assumption.

The paper is organized as follows. In Section 2 we adopt a probabilistic view to outlier detection by taking as a heavy-tailed observation error density the Student-$t$-distribution which can be derived from an infinite mixture of Gaussians approach. In our work we use the multi-layer perceptron (MLP) as nonlinear model. In Section 3 we derive an EM algorithm for estimating the MLP weight vector and the hyperparameters offline. Employing a state-space representation to model the MLP's weight evolution in time we extend the batch algorithm of Section 3 to the online learning case (Section 4). The application of the computationally efficient Fisher scoring algorithm leads to posterior mode weight updates and an online EM-type algorithm for approximate maximum likelihood (ML) estimation of the hyperparameters. In in the last two sections (Section 5 and Section 6) we present experiments and conclusions, respectively.

## 2   THE $t$-DENSITY AS A ROBUST ERROR DENSITY

We assume a nonlinear regression model where for the $t$-th data point the noisy target $y_t \in \mathbf{R}$ is generated as

$$y_t = g(x_t; w_t) + v_t \tag{1}$$

and $x_t \in \mathbf{R}^k$ is a $k$-dimensional known input vector. $g(.; w_t)$ denotes a neural network model characterized by weight vector $w_t \in \mathbf{R}^n$, in our case a multi-layer perceptron (MLP). In the offline case the weight vector $w_t$ is assumed to be a fixed unknown constant vector, i.e. $w_t \equiv w$. Furthermore, we assume that $v_t$ is uncorrelated noise with density $p_{v_t}(.)$. In the offline case, we assume $p_{v_t}(.)$ to be independent of $t$, i.e. $p_{v_t}(.) \equiv p_v(.)$. In the following we assume that $p_v(.)$ is a Student-$t$-density with $\nu$ degrees of freedom with

$$p_v(z) = \mathcal{T}(z | \sigma^2, \nu) = \frac{\Gamma\left(\frac{\nu+1}{2}\right)}{\sigma \sqrt{\pi \nu} \, \Gamma\left(\frac{\nu}{2}\right)} \left(1 + \frac{z^2}{\sigma^2 \nu}\right)^{-\frac{\nu+1}{2}}, \quad \nu, \sigma > 0. \tag{2}$$

It is immediately apparent that for $\nu = 1$ we recover the heavy-tailed Cauchy density. What is not so obvious is that for $\nu \to \infty$ we obtain a Gaussian density. For the derivation of the EM-learning rules in the next section it is important to note that the $t$-denstiy can be thought of as being an infinite mixture of Gaussians of the form

$$\mathcal{T}(z | \sigma^2, \nu) = \int \mathcal{N}(z | 0, \sigma^2 / u) \, p(u) \, du \tag{3}$$

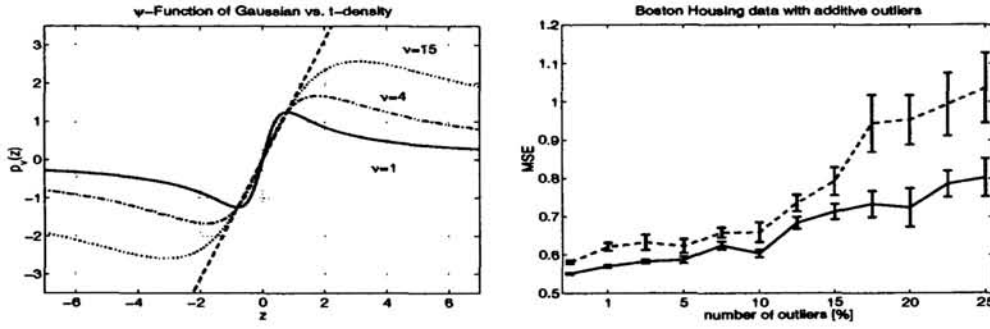

Figure 1: **Left**: $\psi(.)$-functions for the Gaussian density (dashed) and $t$-densities with $\nu = 1, 4, 15$ degrees of freedom. **Right**: MSE on Boston Housing data test set for additive outliers. The dashed line shows results using a Gaussian error measure and the continuous line shows the results using the Student-$t$-distribution as error measure.

where $\mathcal{T}(z|\sigma^2, \nu)$ is the Student-$t$-density with $\nu$ degrees of freedom and width parameter $\sigma^2$, $\mathcal{N}(z|0, \sigma^2/u)$ is a Gaussian density with center 0 and variance $\sigma^2/u$ and $u \sim \chi_\nu^2/\nu$ where $\chi_\nu^2$ is a Chi-square distribution with $\nu$ degrees of freedom evaluated at $u > 0$.

To compare different noise models it is useful to evaluate the "$\psi$-function" defined as (Huber, 1964)

$$\psi(z) = -\partial \log p_v(z)/\partial z \tag{4}$$

i.e. the negative score-function of the noise density. In the case of i.i.d. samples the $\psi$-function reflects the influence of a single measurement on the resulting estimator. Assuming Gaussian measurement errors $p_v(z) = \mathcal{N}(z|0, \sigma^2)$ we derive $\psi(z) = z/\sigma^2$ which means that for $|z| \to \infty$ a single outlier $z$ can have an infinite leverage on the estimator. In contrast, for constructing robust estimators West (1981) states that large outliers should not have *any* influence on the estimator, i.e. $\psi(z) \to 0$ for $|z| \to \infty$. Figure 1 (left) shows $\psi(z)$ for different $\nu$ for the Student-$t$-distribution. It can be seen that the degrees of freedom $\nu$ determine how much weight outliers obtain in influencing the regression. In particular, for finite $\nu$, the influence of outliers with $|z| \to \infty$ approaches zero.

## 3 ROBUST OFFLINE REGRESSION

As stated in Equation (3), the $t$-density can be thought of as being generated as an infinite mixture of Gaussians. Maximum likelihood adaptation of parameters and hyperparameters can therefore be performed using an EM algorithm (Lange *et al.*, 1989). For the $t$-th sample, a complete data point would consist of the triple $(x_t, y_t, u_t)$ of which only the first two are known and $u_t$ is missing.

In the *E-step* we estimate for every data point indexed by $t$

$$\alpha_t = (\nu^{\text{old}} + 1)/(\nu^{\text{old}} + \delta_t) \tag{5}$$

where $\alpha_t = \mathrm{E}[u_t|y_t, x_t]$ is the expected value of the unknown $u_t$ given the available data $(x_t, y_t)$ and where $\delta_t = (y_t - g(x_t; w^{\text{old}}))^2/\sigma^{2, \text{old}}$.

In the *M-step* the weights $w$ and the hyperparameters $\sigma^2$ and $\nu$ are optimized using

$$w^{\text{new}} = \arg \min_w \left\{ \sum_{t=1}^{T} \alpha_t (y_t - g(x_t; w))^2 \right\} \tag{6}$$

$$\sigma^{2,\text{new}} = \frac{1}{T}\sum_{t=1}^{T}\alpha_t\Big[\big(y_t - g(x_t; w^{\text{new}})\big)^2\Big] \tag{7}$$

$$\nu^{\text{new}} = \arg\max_{\nu}\Big\{\frac{T\nu}{2}\log\frac{\nu}{2} - T\log\{\Gamma(\frac{\nu}{2})\}$$

$$+(\frac{\nu}{2}-1)\sum_{t=1}^{T}\beta_t - \frac{\nu}{2}\sum_{t=1}^{T}\alpha_t\Big\} \tag{8}$$

where

$$\beta_t = \text{DG}\big(\frac{\nu^{\text{old}}+1}{2}\big) - \log\big(\frac{1}{2}(\nu^{\text{old}}+\delta_t)\big) \tag{9}$$

with the Digamma function $\text{DG}(z) = \partial\Gamma(z)/\partial z$. Note that the M-step for $\nu$ is a one-dimensional nonlinear optimization problem. Also note that the M-steps for the weights in the MLP reduce to a weighted least squares regression problem in which outliers tend to be weighted down. The exception of course is the Gaussian case with $\nu \to \infty$ in which all terms obtain equal weight.

## 4   ROBUST ONLINE REGRESSION

For robust online regression, we assume that the model Equation (1) is still valid but that $w$ can change over time, i.e. $w \equiv w_t$. In particular we assume that $w_t$ follows a first order random walk with normally distributed increments, i.e.

$$w_t|w_{t-1} \sim \mathcal{N}(w_{t-1}, Q_t) \tag{10}$$

and where $w_0$ is normally distributed with center $a_0$ and covariance $Q_0$. Clearly, due to the nonlinear nature of $g$ and due to the fact that the noise process is non-Gaussian, a fully Bayesian online algorithm — which for the linear case with Gaussian noise can be realized using the Kalman filter — is clearly infeasible.

On the other hand, if we consider data $\mathcal{D} = \{x_t, y_t\}_{t=1}^T$, the negative log-posterior $-\log p(W_T|\mathcal{D})$ of the parameter sequence $W_T = (w_0^\top, \ldots, w_T^\top)^\top$ is up to a normalizing constant

$$-\log p(W_T|\mathcal{D}) \propto -\sum_{t=1}^{T}\log p_v\big(y_t - g(x_t; w_t)\big) + \frac{1}{2}(w_0 - a_0)^\top Q_0^{-1}(w_0 - a_0)$$

$$+\frac{1}{2}\sum_{t=1}^{T}(w_t - w_{t-1})^\top Q_t^{-1}(w_t - w_{t-1}) \tag{11}$$

and can be used as the appropriate cost function to derive the *posterior mode estimate* $W_T^{\text{MAP}}$ for the weight sequence. The two differences to the presentation in the last section are that first, $w_t$ is allowed to change over time and that second, penalty terms, stemming from the prior and the transition density, are included. The penalty terms are penalizing roughness of the weight sequence leading to smooth weight estimates.

A suitable way to determine a stationary point of $-\log p(W_T|\mathcal{D})$, the posterior mode estimate of $W_T$, is to apply *Fisher scoring*. With the current estimate $W_T^{\text{old}}$ we get a better estimate $W_T^{\text{new}} = W_T^{\text{old}} + \eta\gamma$ for the unknown weight sequence $W_T$ where $\gamma$ is the solution of

$$\mathcal{S}(W_T^{\text{old}})\gamma = s(W_T^{\text{old}}) \tag{12}$$

with the negative score function $s(W_T) = -\partial\log p(W_T|\mathcal{D})/\partial W_T$ and the expected information matrix $\mathcal{S}(W_T) = \text{E}[\partial^2\log p(W_T|\mathcal{D})/\partial W_T\partial W_T^\top]$. By applying the ideas given in Fahrmeir & Kaufmann (1991) to robust neural network regression it turns out that solving (12), i.e. to compute the inverse of the expected information matrix, can be performed by

Cholesky decomposition in one forward and backward pass through the set of data $\mathcal{D}$. Note that the expected information matrix is a positive definite block-tridiagonal matrix. The forward-backward steps have to be iterated to obtain the posterior mode estimate $W_T^{\text{MAP}}$ for $W_T$.

For *online posterior mode smoothing*, it is of interest to smooth backwards after each filter step $t$. If Fisher scoring steps are applied sequentially for $t = 1, 2, \ldots$, then the posterior mode smoother at time-step $t - 1$, $W_{t-1}^{\text{MAP}} = (w_{0|t-1}^{\top}, \ldots, w_{t-1|t-1}^{\top})^{\top}$ together with the step-one predictor $w_{t|t-1} = w_{t-1|t-1}$ is a reasonable starting value for obtaining the posterior mode smoother $W_t^{\text{MAP}}$ at time $t$. One can reduce the computational load by limiting the backward pass to a sliding time window, e.g. the last $\tau_t$ time steps, which is reasonable in non-stationary environments for online purposes. Furthermore, if we use the underlying assumption that in most cases a new measurement $y_t$ should not change estimates too drastically then a *single* Fisher scoring step often suffices to obtain the new posterior mode estimate at time $t$. The resulting single Fisher scoring step algorithm with lookback parameter $\tau_t$ has in fact just one additional line of code involving simple matrix manipulations compared to online Kalman smoothing and is given here in pseudo-code. Details about the algorithm and a full description can be found in Briegel & Tresp (1999).

*Online single Fisher scoring step algorithm (pseudo-code)*

> *for $t = 1, 2, \ldots$ repeat the following four steps*:
> - *Evaluate the step-one predictor $w_{t|t-1}$.*
> - *Perform the forward recursions for $s = t - \tau_t, \ldots, t$.*
> - *New data point $(x_t, y_t)$ arrives: evaluate the corrector step $w_{t|t}$.*
> - *Perform the backward smoothing recursions $w_{s-1|t}$ for $s = t, \ldots, t - \tau_t$.*

For the adaptation of the parameters in the $t$-distribution, we apply results from Fahrmeir & Künstler (1999) to our nonlinear assumptions and use an online EM-type algorithm for approximate maximum likelihood estimation of the hyperparameters $\nu_t$ and $\sigma_t^2$. We assume the scale factors $\sigma_t^2$ and the degrees of freedom $\nu_t$ being fixed quantities in a certain time window of length $\tilde{\tau}_t$, e.g. $\sigma_t^2 = \sigma^2, \nu_t = \nu, t \in \{t - \tilde{\tau}_t, t\}$. For deriving online EM update equations we treat the weight sequence $w_t$ together with the mixing variables $u_t$ as missing. By linear Taylor series expansion of $g(.; w_s)$ about the Fisher scoring solutions $w_{s|t}$ and by approximating posterior expectations $\mathrm{E}[w_s|\mathcal{D}]$ with posterior modes $w_{s|t}, s \in \{t - \tilde{\tau}_t, t\}$ and posterior covariances $\mathrm{cov}[w_s|\mathcal{D}]$ with curvatures $\Sigma_{s|t} = \mathrm{E}[(w_s - w_{s|t})(w_s - w_{s|t})^{\top}|\mathcal{D}]$ in the E-step, a somewhat lengthy derivation results in approximate maximum likelihood update rules for $\sigma^2$ and $\nu$ similar to those given in Section 3. Details about the online EM-type algorithm can be found in Briegel & Tresp (1999).

## 5  EXPERIMENTS

**1. Experiment: Real World Data Sets.** In the first experiment we tested if the Student-$t$-distribution is a useful error measure for real-world data sets. In training, the Student-$t$-distribution was used and both, the degrees of freedom $\nu$ and the width parameter $\sigma^2$ were adapted using the EM update rules from Section 3. Each experiment was repeated 50 times with different divisions into training and test data. As a comparison we trained the neural networks to minimize the squared error cost function (including an optimized weight decay term). On the test data set we evaluated the performance using a squared error cost function. Table 1 provides some experimental parameters and gives the test set performance based on the 50 repetitions of the experiments. The additional explained variance is defined as [in percent] $100 \times (1 - \text{MSPE}_{\mathcal{T}}/\text{MSPE}_{\mathcal{N}})$ where $\text{MSPE}_{\mathcal{T}}$ is the mean squared prediction error using the $t$-distribution and $\text{MSPE}_{\mathcal{N}}$ is the mean squared prediction error using the Gaussian error measure. Furthermore we supply the standard

Table 1: Experimental parameters and test set performance on real world data sets.

| Data Set | # Inputs/Hidden | Training | Test | Add.Exp.Var. [%] | Std. [%] |
|---|---|---|---|---|---|
| Boston Housing | (13/6) | 400 | 106 | 4.2 | 0.93 |
| Sunspot | (12/7) | 221 | 47 | 5.3 | 0.67 |
| Fraser River | (12/7) | 600 | 334 | 5.4 | 0.75 |

error based on the 50 experiments. In all three experiments the networks optimized with the $t$-distribution as noise model were 4-5% better than the networks optimized using the Gaussian as noise model and in all experiments the improvements were significant based on the paired $t$-test with a significance level of 1%. The results show clearly that the additional free parameter in the Student-$t$-distribution does not lead to overfitting but is used in a sensible way by the system to value down the influence of extreme target values. Figure 2 shows the normal probability plots. Clearly visible is the derivation from the Gaussian distribution for extreme target values. We also like to remark that we did not apply any preselection process in choosing the particular data sets which indicates that non-Gaussian noise seems to be the rule rather than the exception for real world data sets.

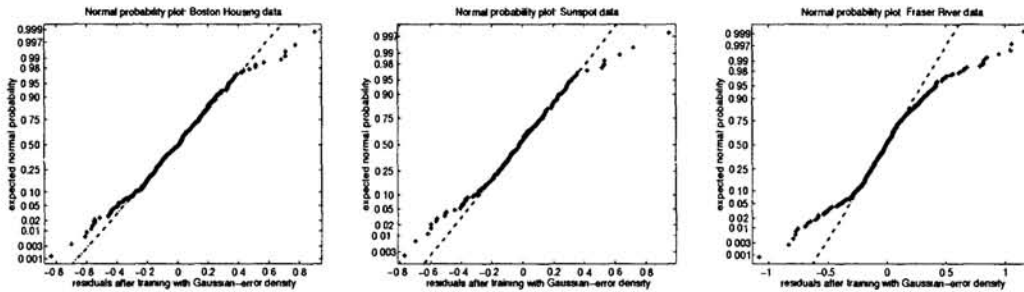

Figure 2: Normal probability plots of the three training data sets after learning with the Gaussian error measure. The dashed line show the expected normal probabilities. The plots show clearly that the residuals follow a more heavy-tailed distribution than the normal distribution.

**2. Experiment: Outliers.** In the second experiment we wanted to test how our approach deals with outliers which are artificially added to the data set. We started with the Boston housing data set and divided it into training and test data. We then randomly selected a subset of the training data set (between 0.5% and 25%) and added to the targets a uniformly generated real number in the interval $[-5, 5]$. Figure 1 (right) shows the mean squared error on the test set for different percentages of added outliers. The error bars are derived from 20 repetitions of the experiment with different divisions into training and test set. It is apparent that the approach using the $t$-distribution is consistently better than the network which was trained based on a Gaussian noise assumption.

**3. Experiment: Online Learning.** In the third experiment we examined the use of the $t$-distribution in online learning. Data were generated from a nonlinear map $y = 0.6x^2 + b\sin(6x) - 1$ where $b = -0.75, -0.4, -0.1, 0.25$ for the first, second, third and fourth set of 150 data points, respectively. Gaussian noise with variance 0.2 was added and for training, a MLP with 4 hidden units was used. In the first experiment we compare the performance of the EKF algorithm with our single Fisher scoring step algorithm. Figure 3 (left) shows that our algorithm converges faster to the correct map and also handles the transition in the model (parameter $b$) much better than the EKF. In the second experiment with a probability of 10% outliers uniformly drawn from the interval $[-5, 5]$ were added to the targets. Figure 3 (middle) shows that the single Fisher scoring step algorithm using the

$t$-distribution is consistently better than the same algorithm using a Gaussian noise model and the EKF. The two plots on the right in Figure 3 compare the nonlinear maps learned after 150 and 600 time steps, respectively.

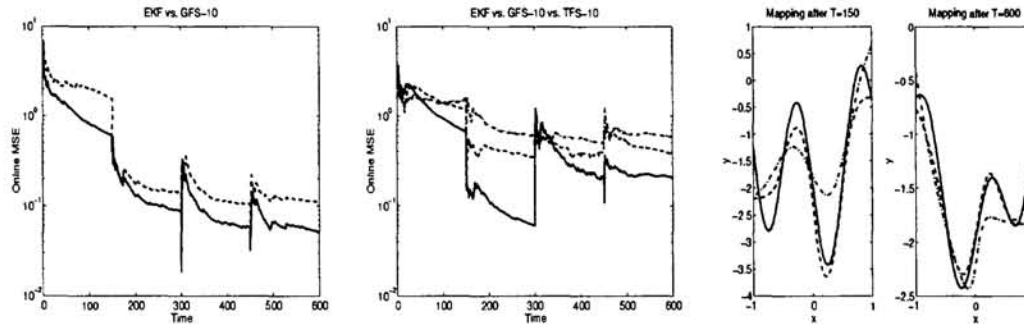

Figure 3: **Left & Middle**: Online MSE over each of the 4 sets of training data. On the left we compare extended Kalman filtering (EKF) (dashed) with the single Fisher scoring step algorithm with $\tau_t = 10$ (GFS-10) (continuous) for additive Gaussian noise. The second figure shows EKF (dashed-dotted), Fisher scoring with Gaussian error noise (GFS-10) (dashed) and $t$-distributed error noise (TFS-10) (continuous), respectively for data with additive outliers. **Right**: True map (continuous), EKF learned map (dashed-dotted) and TFS-10 map (dashed) after $T = 150$ and $T = 600$ (data sets with additive outliers).

## 6  CONCLUSIONS

We have introduced the Student-$t$-distribution to replace the standard Gaussian noise assumption in nonlinear regression. Learning is based on an EM algorithm which estimates both the scaling parameters and the degrees of freedom of the $t$-distribution. Our results show that using the Student-$t$-distribution as noise model leads to 4-5% better test errors than using the Gaussian noise assumption on real world data set. This result seems to indicate that non-Gaussian noise is the rule rather than the exception and that extreme target values should in general be weighted down. Dealing with outliers is particularly important for online tasks in which outliers can lead to instability in the adaptation process. We introduced a new online learning algorithm using the $t$-distribution which leads to better and more stable results if compared to the extended Kalman filter.

## Footnotes

*Now with McKinsey & Company, Inc.

### References

Briegel, T. and Tresp, V. (1999) *Dynamic Neural Regression Models*, Discussion Paper, Seminar für Statistik, Ludwig Maximilians Universität München.

de Freitas, N., Doucet, A. and Niranjan, M. (1998) *Sequential Inference and Learning*, NIPS*98 Workshop, Breckenridge, CO.

Fahrmeir, L. and Kaufmann, H. (1991) *On Kalman Filtering, Posterior Mode Estimation and Fisher Scoring in Dynamic Exponential Family Regression*, Metrika 38, pp. 37-60.

Fahrmeir, L. and Künstler, R. (1999) *Penalized likelihood smoothing in robust state space models*, Metrika 49, pp. 173-191.

Huber, P.J. (1964) *Robust Estimation of Location Parameter*, Annals of Mathematical Statistics 35, pp. 73-101.

Lange, K., Little, L., Taylor, J. (1989) *Robust Statistical Modeling Using the t-Distribution*, JASA 84, pp. 881-896.

Meinhold, R. and Singpurwalla, N. (1989) *Robustification of Kalman Filter Models*, JASA 84, pp. 470-496.

Rousseeuw, P. and Leroy, A. (1987) *Robust Regression and Outlier Detection*, John Wiley & Sons.

West, M. (1981) *Robust Sequential Approximate Bayesian Estimation*, JRSS B 43, pp. 157-166.
